# BIT - SERIAL NEURAL NETWORKS

Alan F. Murray, Anthony V. W. Smith and Zoe F. Butler.
Department of Electrical Engineering, University of Edinburgh,
The King's Buildings, Mayfield Road, Edinburgh,
Scotland, EH9 3JL.

## ABSTRACT

A bit - serial VLSI neural network is described from an initial architecture for a synapse array through to silicon layout and board design. The issues surrounding bit - serial computation, and analog/digital arithmetic are discussed and the parallel development of a hybrid analog/digital neural network is outlined. Learning and recall capabilities are reported for the bit - serial network along with a projected specification for a 64 - neuron, bit - serial board operating at 20 MHz. This technique is extended to a 256 ($256^2$ synapses) network with an update time of 3ms, using a "paging" technique to time - multiplex calculations through the synapse array.

## 1. INTRODUCTION

The functions a synthetic neural network may aspire to mimic are the ability to consider many solutions simultaneously, an ability to work with corrupted data and a natural fault tolerance. This arises from the parallelism and distributed knowledge representation which gives rise to gentle degradation as faults appear. These functions are attractive to implementation in VLSI and WSI. For example, the natural fault - tolerance could be useful in silicon wafers with imperfect yield, where the network degradation is approximately proportional to the non-functioning silicon area.

To cast neural networks in engineering language, a neuron is a state machine that is either "on" or "off", which in general assumes intermediate states as it switches smoothly between these extrema. The synapses *weighting* the signals from a transmitting neuron such that it is more or less excitatory or inhibitory to the receiving neuron. The set of synaptic weights determines the stable states and represents the learned information in a system.

The neural state, $V_i$, is related to the total neural activity stimulated by inputs to the neuron through an *activation function*, F. Neural activity is the level of excitation of the neuron and the activation is the way it reacts in a response to a change in activation. The neural output state at time t, $V_i^t$, is related to $x_i^t$ by

$$V_i^t = F(x_i^t) \tag{1}$$

The activation function is a "squashing" function ensuring that (say) $V_i$ is 1 when $x_i$ is large and -1 when $x_i$ is small. The neural update function is therefore straightforward:

$$x_i^{t+1} = x_i^t \ \ldots + \delta \sum_{j=0}^{j=n-1} T_{ij} V_j^t \tag{2}$$

where $\delta$ represents the rate of change of neural activity, $T_{ij}$ is the synaptic weight and n is the number of terms giving an n - neuron array [1].

Although the *neural* function is simple enough, in a totally interconnected n - neuron network there are $n^2$ synapses requiring $n^2$ multiplications and summations and

a large number of interconnects. The challenge in VLSI is therefore to design a simple, compact synapse that can be repeated to build a VLSI neural network with manageable interconnect. In a network with fixed functionality, this is relatively straightforward. If the network is to be able to learn, however, the synaptic weights must be programmable, and therefore more complicated.

## 2. DESIGNING A NEURAL NETWORK IN VLSI

There are fundamentally two approaches to implementing any function in silicon - digital and analog. Each technique has its advantages and disadvantages, and these are listed below, along with the merits and demerits of bit - serial architectures in digital (synchronous) systems.

**Digital vs. analog:** The primary advantage of digital design for a synapse array is that digital memory is well understood, and can be incorporated easily. Learning networks are therefore possible without recourse to unusual techniques or technologies. Other strengths of a digital approach are that design techniques are advanced, automated and well understood and noise immunity and computational speed can be high. Unattractive features are that digital circuits of this complexity need to be synchronous and all states and activities are quantised, while real neural networks are asynchronous and unquantised. Furthermore, digital multipliers occupy a large silicon area, giving a low synapse count on a single chip.

The advantages of analog circuitry are that asynchronous behaviour and smooth neural activation are automatic. Circuit elements can be small, but noise immunity is relatively low and arbitrarily high precision is not possible. Most importantly, no reliable analog, non - volatile memory technology is as yet readily available. For this reason, learning networks lend themselves more naturally to digital design and implementation.

Several groups are developing neural chips and boards, and the following listing does not pretend to be exhaustive. It is included, rather, to indicate the spread of activity in this field. Analog techniques have been used to build resistor / operational amplifier networks [2, 3] similar to those proposed by Hopfield and Tank [4]. A large group at Caltech is developing networks implementing early vision and auditory processing functions using the intrinsic nonlinearities of MOS transistors in the subthreshold regime [5, 6]. The problem of implementing analog networks with electrically programmable synapses has been addressed using CCD/MNOS technology [7]. Finally, Garth [8] is developing a digital neural accelerator board ("Netsim") that is effectively a fast SIMD processor with supporting memory and communications chips.

**Bit - serial vs. bit - parallel:** Bit - serial arithmetic and communication is efficient for computational processes, allowing good communication within and between VLSI chips and tightly pipelined arithmetic structures. It is ideal for neural networks as it minimises the interconnect requirement by eliminating multi - wire busses. Although a bit - parallel design would be free from computational latency (delay between input and output), pipelining makes optimal use of the high bit - rates possible in serial systems, and makes for efficient circuit usage.

### 2.1 An asynchronous pulse stream VLSI neural network:

In addition to the digital system that forms the substance of this paper, we are developing a hybrid analog/digital network family. This work is outlined here, and has been reported in greater detail elsewhere [9, 10, 11]. The generic (logical and layout) architecture of a single network of n totally *interconnected* neurons is shown

schematically in figure 1. Neurons are represented by circles, which signal their states, $V_i$ upward into a matrix of synaptic operators. The state signals are connected to a $n$ - bit horizontal bus running through the synaptic array, with a connection to each synaptic operator in every column. All columns have n operators (denoted by squares) and each operator adds its synaptic contribution, $T_{ij}V_j$, to the running total of activity for the neuron $i$ at the foot of the column. The synaptic function is therefore to *multiply* the signalling neuron state, $V_j$, by the synaptic weight, $T_{ij}$, and to *add* this product to the running total. This architecture is common to both the bit - serial and pulse - stream networks.

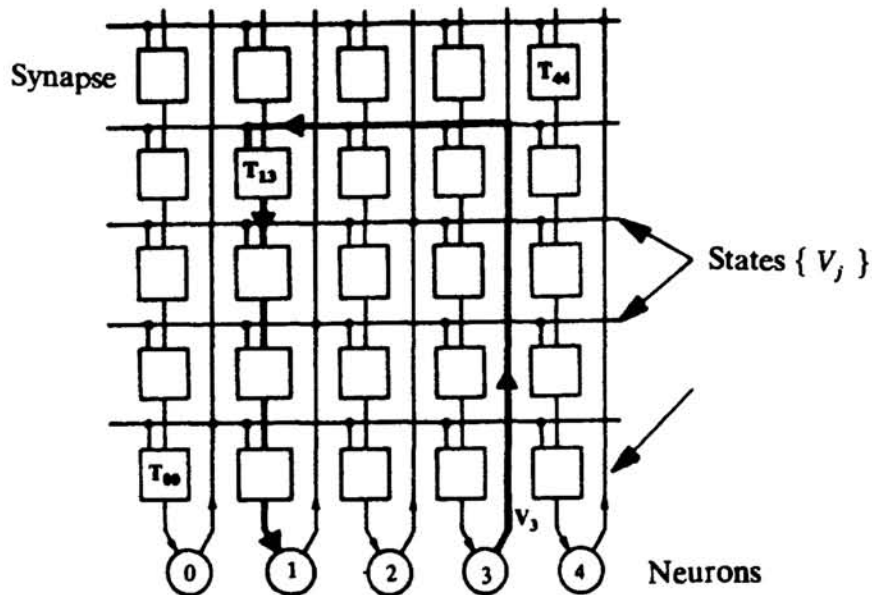

**Figure 1.** *Generic architecture for a network of n totally interconnected neurons.*

This type of architecture has many attractions for implementation in 2 - dimensional silicon as the summation $\sum_{j=0}^{j=n-1} T_{ij}V_j$ is distributed in space. The interconnect requirement (n inputs to each neuron) is therefore distributed through a column, reducing the need for long - range wiring. The architecture is modular, regular and can be easily expanded.

In the hybrid analog/digital system, the circuitry uses a "pulse stream" signalling method similar to that in a natural neural system. Neurons indicate their state by the presence or absence of pulses on their outputs, and synaptic weighting is achieved by time - chopping the presynaptic pulse stream prior to adding it to the postsynaptic activity summation. It is therefore asynchronous and imposes no fundamental limitations on the activation or neural state. Figure 2 shows the pulse stream mechanism in more detail. The synaptic weight is stored in digital memory local to the operator. Each synaptic operator has an excitatory and inhibitory pulse stream input and output. The resultant product of a synaptic operation, $T_{ij}V_j$, is added to the running total propagating down either the excitatory or inhibitory channel. One binary bit (the MSBit) of the stored $T_{ij}$ determines whether the contribution is excitatory or inhibitory.

The incoming excitatory and inhibitory pulse stream inputs to a neuron are integrated to give a neural activation potential that varies smoothly from 0 to 5 V. This potential controls a feedback loop with an odd number of logic inversions and

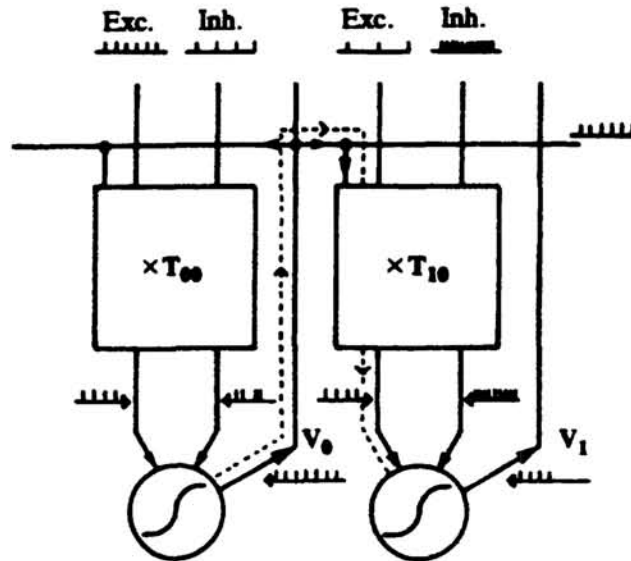

**Figure 2.** *Pulse stream arithmetic. Neurons are denoted by $\bigcirc$ and synaptic operators by $\square$.*

thus forms a switched "ring - oscillator". If the inhibitory input dominates, the feedback loop is broken. If excitatory spikes subsequently dominate at the input, the neural activity rises to 5V and the feedback loop oscillates with a period determined by a delay around the loop. The resultant periodic waveform is then converted to a series of voltage spikes, whose pulse rate represents the neural state, $V_i$. Interestingly, a not dissimilar technique is reported elsewhere in this volume, although the synapse function is executed differently [12].

## 3. A 5 - STATE BIT - SERIAL NEURAL NETWORK

The overall architecture of the 5 - state bit - serial neural network is identical to that of the pulse stream network. It is an array of $n^2$ interconnected synchronous synaptic operators, and whereas the pulse stream method allowed $V_j$ to assume all values between "off" and "on", the 5 - state network $V_j$ is constrained to 0, $\pm 0.5$ or $\pm 1$. The resultant activation function is shown in Figure 3. Full digital multiplication is costly in silicon area, but multiplication of $T_{ij}$ by $V_j = 0.5$ merely requires the synaptic weight to be right - shifted by 1 bit. Similarly, multiplication by 0.25 involves a further right - shift of $T_{ij}$, and multiplication by 0.0 is trivially easy. $V_j < 0$ is not problematic, as a switchable adder/subtractor is not much more complex than an adder. Five neural states are therefore feasible with circuitry that is only slightly more complex than a simple serial adder. The neural state expands from a 1 bit to a 3 bit (5 - state) representation, where the bits represent "add/subtract?", "shift?" and "multiply by 0?".

Figure 4 shows part of the synaptic array. Each synaptic operator includes an 8 bit shift register memory block holding the synaptic weight, $T_{ij}$. A 3 bit bus for the 5 neural states runs horizontally above each synaptic row. Single phase dynamic CMOS has been used with a clock frequency in excess of 20 MHz [13]. Details of a synaptic operator are shown in figure 5. The synaptic weight $T_{ij}$ cycles around the shift register and the neural state $V_j$ is present on the state bus. During the first clock cycle, the synaptic weight is multiplied by the neural state and during the second, the most significant bit (MSBit) of the resultant $T_{ij}V_j$ is sign - extended for

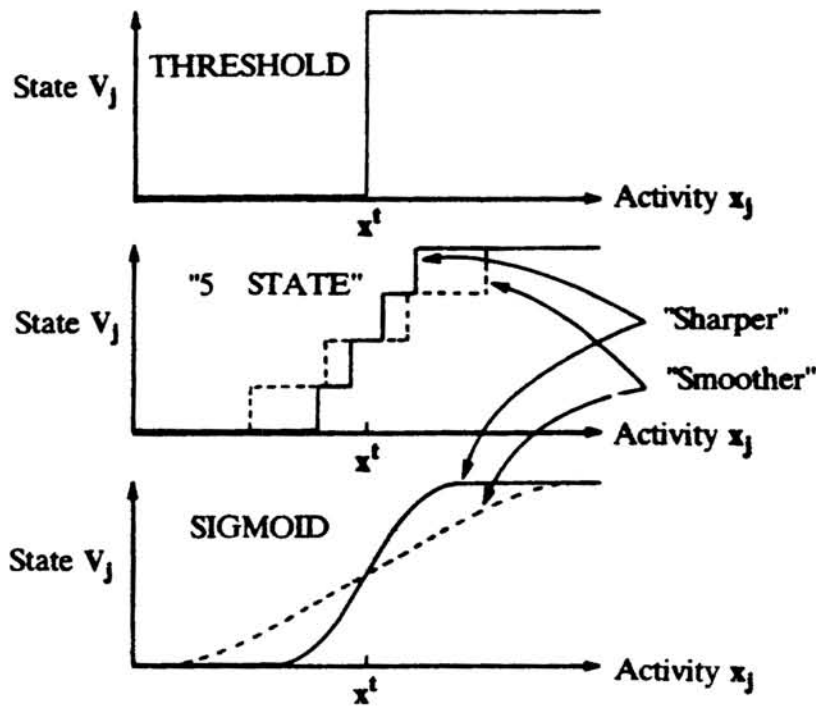

**Figure 3.** *"Hard - threshold", 5 - state and sigmoid activation functions.*

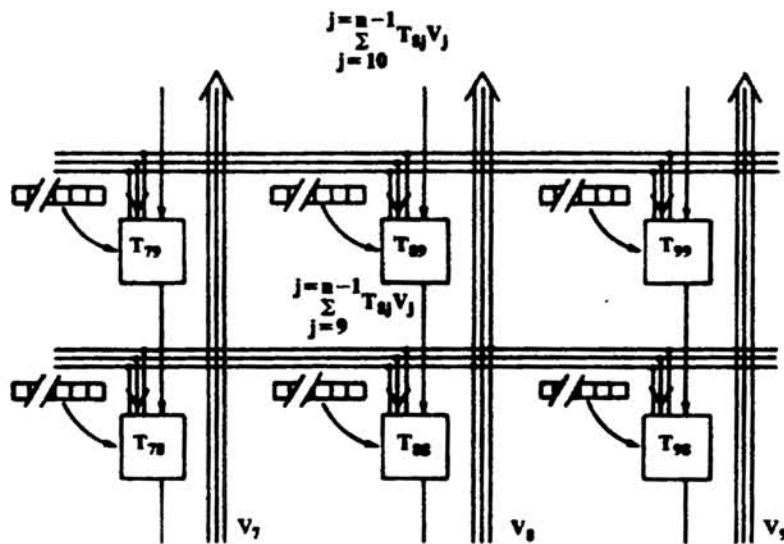

**Figure 4.** *Section of the synaptic array of the 5 - state activation function neural network.*

8 bits to allow for word growth in the running summation. A least significant bit (LSBit) signal running down the synaptic columns indicates the arrival of the LSBit of the $x_i$ running total. If the neural state is $\pm 0.5$ the synaptic weight is right shifted by 1 bit and then added to or subtracted from the running total. A multiplication of $\pm 1$ adds or subtracts the weight from the total and multiplication by 0

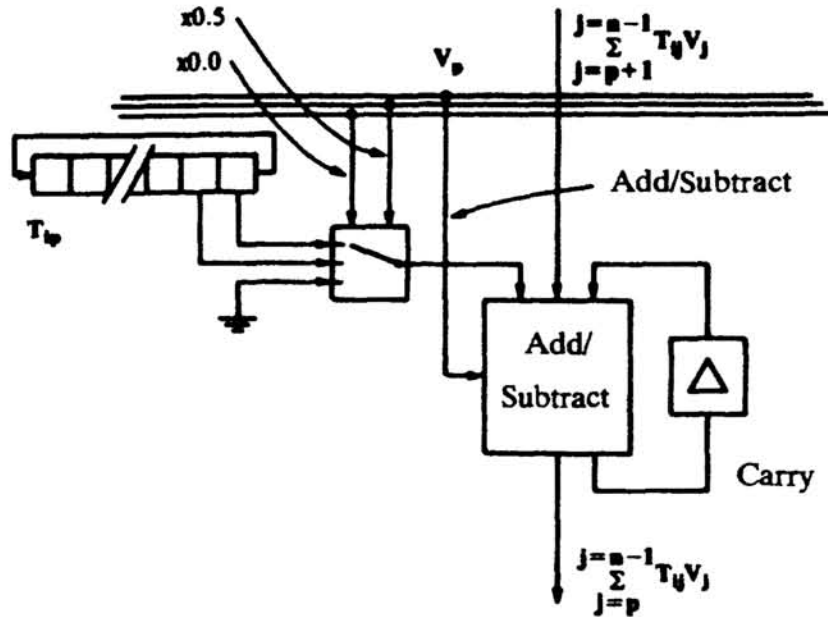

**Figure 5.** *The synaptic operator with a 5 - state activation function.*

does not alter the running summation.

The final summation at the foot of the column is thresholded externally according to the 5 - state activation function in figure 3. As the neuron activity $x_j$, increases through a threshold value $x_t$, ideal sigmoidal activation represents a smooth switch of neural state from -1 to 1. The 5 - state "staircase" function gives a superficially much better approximation to the sigmoid form than a (much simpler to implement) threshold function. The sharpness of the transition can be controlled to "tune" the neural dynamics for learning and computation. The control parameter is referred to as temperature by analogy with statistical functions with this sigmoidal form. High "temperature" gives a smoother staircase and sigmoid, while a temperature of 0 reduces both to the "Hopfield" - like threshold function. The effects of temperature on both learning and recall for the threshold and 5 - state activation options are discussed in section 4.

## 4. LEARNING AND RECALL WITH VLSI CONSTRAINTS

Before implementing the reduced - arithmetic network in VLSI, simulation experiments were conducted to verify that the 5 - state model represented a worthwhile enhancement over simple threshold activation. The "benchmark" problem was chosen for its ubiquitousness, rather than for its intrinsic value. The implications for learning and recall of the 5 - state model, the threshold (2 - state) model and smooth sigmoidal activation ( $\infty$ - state) were compared at varying temperatures with a restricted dynamic range for the weights $T_{ij}$. In each simulation a totally interconnected 64 node network attempted to learn 32 random patterns using the delta rule learning algorithm (see for example [14]). Each pattern was then corrupted with 25% noise and recall attempted to probe the content addressable memory properties under the three different activation options.

During learning, individual weights can become large (positive or negative). When weights are "driven" beyond the maximum value in a hardware implementation,

which is determined by the size of the synaptic weight blocks, some limiting mechanism must be introduced. For example, with eight bit weight registers, the limitation is $-128 \leq T_{ij} \leq 127$. With integer weights, this can be seen to be a problem of *dynamic range*, where it is the relationship between the smallest possible weight ($\pm 1$) and the largest ($+127/-128$) that is the issue.

**Results:** Fig. 6 shows examples of the results obtained, studying *learning* using 5 - state activation at different temperatures, and recall using both 5 - state and threshold activation. At temperature T=0, the 5 - state and threshold models are degenerate, and the results identical. Increasing smoothness of activation (temperature) during learning improves the *quality* of learning regardless of the activation function used in recall, as more patterns are recognised successfully. Using 5 - state activation in recall is more effective than simple threshold activation. The effect of dynamic range restrictions can be assessed from the horizontal axis, where $T_{ij}^{max}$ is shown. The results from these and many other experiments may be summarised as follows:-

**5 - State activation vs. threshold:**

1) Learning with 5 - state activation was protracted over the threshold activation, as *binary* patterns were being learnt, and the inclusion of intermediate values added extra degrees of freedom.

2) Weight sets learnt using the 5 - state activation function were "better" than those learnt via threshold activation, as the recall properties of both 5 - state and threshold networks using such a weight set were more robust against noise.

3) Full sigmoidal activation was better than 5 - state, but the enhancement was less significant than that incurred by moving from threshold → 5 - state. This suggests that the law of diminishing returns applies to addition of levels to the neural state $V_j$. This issue has been studied mathematically [15], with results that agree qualitatively with ours.

**Weight Saturation:**

Three methods were tried to deal with weight saturation. Firstly, inclusion of a decay, or "forgetting" term was included in the learning cycle [1]. It is our view that this technique *can* produce the desired weight limiting property, but in the time available for experiments, we were unable to "tune" the rate of decay sufficiently well to confirm it. Renormalisation of the weights (division to bring large weights back into the dynamic range) was very unsuccessful, suggesting that information distributed throughout the numerically small weights was being destroyed. Finally, the weights were allowed to "clip" (ie any weight outside the dynamic range was set to the maximum allowed value). This method proved very successful, as the learning algorithm adjusted the weights over which it still had control to compensate for the saturation effect. It is interesting to note that other experiments have indicated that Hopfield nets can "forget" in a different way, under different learning control, giving preference to recently acquired memories [16]. The results from the saturation experiments were:-

1) For the 32 pattern/64 node problem, integer weights with a dynamic range greater than $\pm 30$ were necessary to give enough storage capability.

2) For weights with maximum values $T_{ij}^{max} = 50 \rightarrow 70$, "clipping" occurs, but network performance is not seriously degraded over that with an unrestricted weight set.

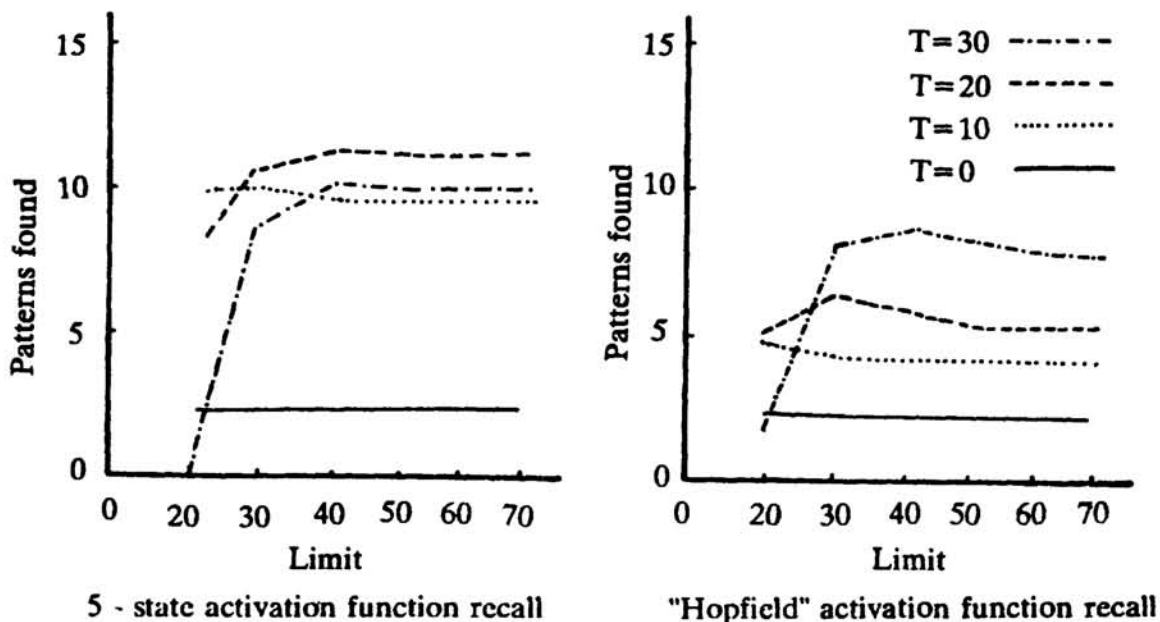

**Figure 6.** *Recall of patterns learned with the 5 - state activation function and subsequently restored using the 5-state and the hard - threshold activation functions.*
*T is the "temperature", or smoothness of the activation function, and "limit" the value of $T_{ij}^{max}$.*

These results showed that the 5 - state model was worthy of implementation as a VLSI neural board, and suggested that 8 - bit weights were sufficient.

## 5. PROJECTED SPECIFICATION OF A HARDWARE NEURAL BOARD

The specification of a 64 neuron board is given here, using a 5 - state bit - serial 64 x 64 synapse array with a derated clock speed of 20 MHz. The synaptic weights are 8 bit words and the word length of the running summation $x_i$ is 16 bits to allow for growth. A 64 synapse column has a computational latency of 80 clock cycles or bits, giving an update time of 4μs for the network. The time to load the weights into the array is limited to 60μs by the supporting RAM, with an access time of 120ns. These load and update times mean that the network is executing $1 \times 10^9$ operations/second, where one operation is $\pm T_{ij}V_j$. This is much faster than a natural neural network, and much faster than is necessary in a hardware accelerator. We have therefore developed a "paging" architecture, that effectively "trades - off" some of this excessive speed against increased network size.

A **"moving - patch" neural board:** An array of the 5 - state synapses is currently being fabricated as a VLSI integrated circuit. The shift registers and the adder/subtractor for each synapse occupy a disappointingly large silicon area, allowing only a 3 x 9 synaptic array. To achieve a suitable size neural network from this array, several chips need to be included on a board with memory and control circuitry. The "moving patch" concept is shown in figure 7, where a small array of synapses is passed over a much larger n x n synaptic array.

Each time the array is "moved" to represent another set of synapses, new weights must be loaded into it. For example, the first set of weights will be $T_{11} \dots T_{ij} \dots T_{21} \dots T_{2j}$ to $T_{jj}$, the second set $T_{j+1,1}$ to $T_{ss}$ etc.. The final weight to be loaded will be

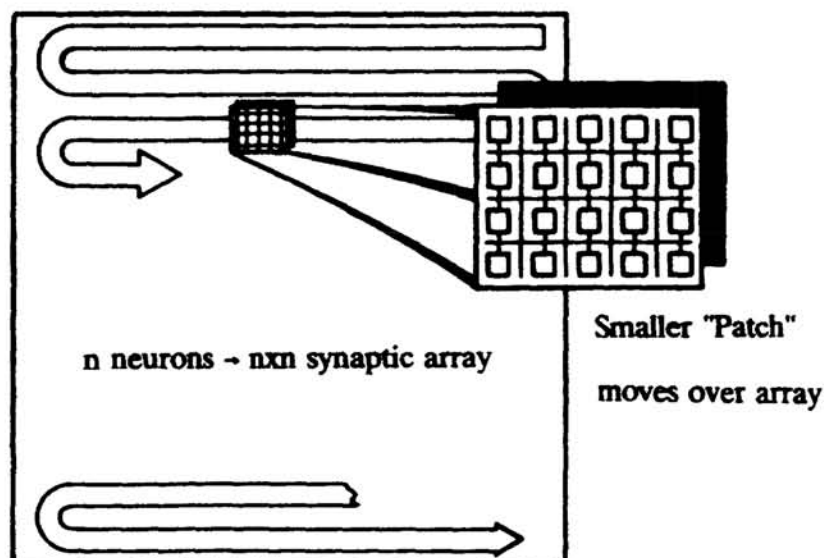

**Figure 7.** *The "moving patch" concept, passing a small synaptic "patch" over a larger nxn synapse array.*

$T_{nn}$. Static, off - the - shelf RAM is used to store the weights and the whole operation is pipelined for maximum efficiency. Figure 8 shows the board level design for the network.

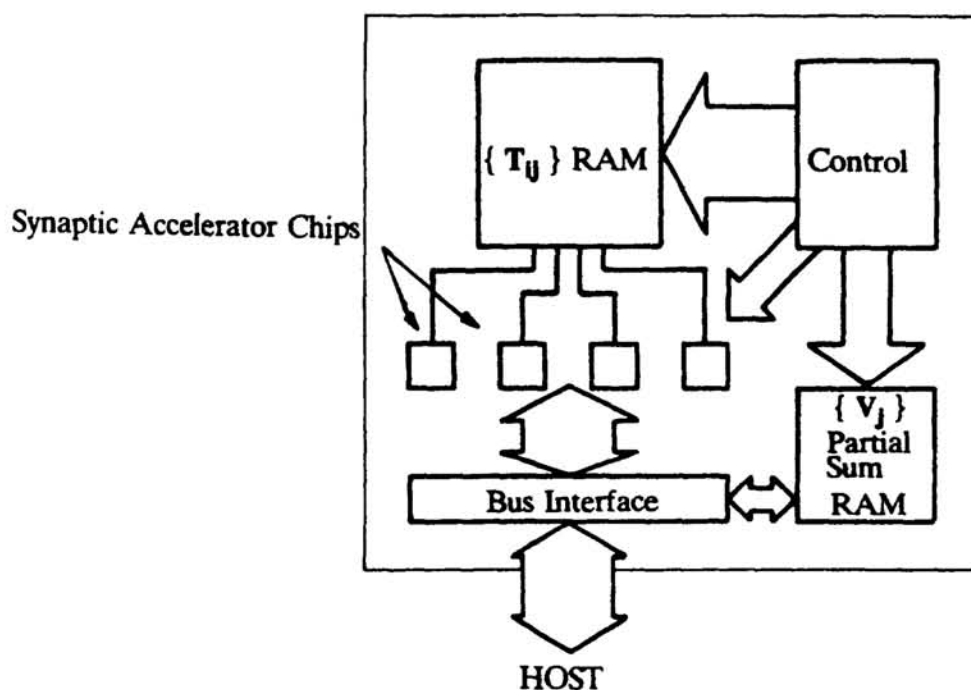

**Figure 8.** *A "moving patch" neural network board.*

The small "patch" that moves around the array to give n neurons comprises 4 VLSI synaptic accelerator chips to give a 6 x 18 synaptic array. The number of neurons to be simulated is 256 and the weights for these are stored in 0.5 Mb of RAM with a load time of 8ms. For each "patch" movement, the partial running summation ⋲

calculated for each column, is stored in a separate RAM until it is required to be added into the next appropriate summation. The update time for the board is 3ms giving $2 \times 10^7$ operations/second. This is slower than the 64 neuron specification, but the network is 16 times larger, as the arithmetic elements are being used more efficiently. To achieve a network of greater than 256 neurons, more RAM is required to store the weights. The network is then slower unless a larger number of accelerator chips is used to give a larger moving "patch".

## 6. CONCLUSIONS

A strategy and design method has been given for the construction of bit - serial VLSI neural network chips and circuit boards. Bit - serial arithmetic, coupled to a reduced arithmetic style, enhances the level of integration possible beyond more conventional digital, bit - parallel schemes. The restrictions imposed on both synaptic weight size and arithmetic precision by VLSI constraints have been examined and shown to be tolerable, using the associative memory problem as a test.

While we believe our digital approach to represent a good compromise between arithmetic accuracy and circuit complexity, we acknowledge that the level of integration is disappointingly low. It is our belief that, while digital approaches may be interesting and useful in the medium term, essentially as hardware accelerators for neural simulations, analog techniques represent the best ultimate option in 2 - dimensional silicon. To this end, we are currently pursuing techniques for analog pseudo - static memory, using standard CMOS technology. In any event, the full development of a nonvolatile analog memory technology, such as the MNOS technique [7], is key to the long - term future of VLSI neural nets that can learn.

## 7. ACKNOWLEDGEMENTS

The authors acknowledge the support of the Science and Engineering Research Council (UK) in the execution of this work.

**References**

1.  S. Grossberg, "Some Physiological and Biochemical Consequences of Psychological Postulates," *Proc. Natl. Acad. Sci. USA*, vol. 60, pp. 758 - 765, 1968.

2.  H. P. Graf, L. D. Jackel, R. E. Howard, B. Straughn, J. S. Denker, W. Hubbard, D. M. Tennant, and D. Schwartz, "VLSI Implementation of a Neural Network Memory with Several Hundreds of Neurons," *Proc. AIP Conference on Neural Networks for Computing, Snowbird*, pp. 182 - 187, 1986.

3.  W. S. Mackie, H. P. Graf, and J. S. Denker, "Microelectronic Implementation of Connectionist Neural Network Models," *IEEE Conference on Neural Information Processing Systems, Denver*, 1987.

4.  J. J. Hopfield and D. W. Tank, "Neural" Computation of Decisions in Optimisation Problems," *Biol. Cybern.*, vol. 52, pp. 141 - 152, 1985.

5.  M. A. Sivilotti, M. A. Mahowald, and C. A. Mead, *Real - Time Visual Computations Using Analog CMOS Processing Arrays*, 1987. To be published

6.  C. A. Mead, "Networks for Real - Time Sensory Processing," *IEEE Conference on Neural Information Processing Systems, Denver*, 1987.

7.  J. P. Sage, K. Thompson, and R. S. Withers, "An Artificial Neural Network Integrated Circuit Based on MNOS/CCD Principles," *Proc. AIP Conference on Neural Networks for Computing, Snowbird*, pp. 381 - 385, 1986.

8.  S. C. J. Garth, "A Chipset for High Speed Simulation of Neural Network Systems," *IEEE Conference on Neural Networks, San Diego*, 1987.

9.  A. F. Murray and A. V. W. Smith, "A Novel Computational and Signalling Method for VLSI Neural Networks," *European Solid State Circuits Conference* , 1987.

10. A. F. Murray and A. J. W. Smith, "Asynchronous Arithmetic for VLSI Neural Systems," *Electronics Letters*, vol. 23, no. 12, p. 642, June, 1987.

11. A. F. Murray and A. V. W. Smith, "Asynchronous VLSI Neural Networks using Pulse Stream Arithmetic," *IEEE Journal of Solid-State Circuits and Systems*, 1988. To be published

12. M. E. Gaspar, "Pulsed Neural Networks : Hardware, Software and the Hopfield A/D Converter Example," *IEEE Conference on Neural Information Processing Systems, Denver*, 1987.

13. M. S. McGregor, P. B. Denyer, and A. F. Murray, "A Single - Phase Clocking Scheme for CMOS VLSI," *Advanced Research in VLSI : Proceedings of the 1987 Stanford Conference*, 1987.

14. D. E. Rumelhart, G. E. Hinton, and R. J. Williams, "Learning Internal Representations by Error Propagation," *Parallel Distributed Processing : Explorations in the Microstructure of Cognition*, vol. 1, pp. 318 - 362, 1986.

15. M. Fleisher and E. Levin, "The Hopfiled Model with Multilevel Neurons Models," *IEEE Conference on Neural Information Processing Systems, Denver*, 1987.

16. G. Parisi, "A Memory that Forgets," *J. Phys. A : Math. Gen.*, vol. 19, pp. L617 - L620, 1986.
